# A Neural Network Autoassociator for Induction Motor Failure Prediction

**Thomas Petsche, Angelo Marcantonio, Christian Darken,**
**Stephen J. Hanson, Gary M. Kuhn and Iwan Santoso**
[PETSCHE, ANGELO, DARKEN, JOSE, GMK, NIS]@SCR.SIEMENS.COM
Siemens Corporate Research, Inc.
755 College Road East
Princeton, NJ 08853

## Abstract

We present results on the use of neural network based autoassociators which act as novelty or anomaly detectors to detect imminent motor failures. The autoassociator is trained to reconstruct spectra obtained from the healthy motor. In laboratory tests, we have demonstrated that the trained autoassociator has a small reconstruction error on measurements recorded from healthy motors but a larger error on those recorded from a motor with a fault. We have designed and built a motor monitoring system using an autoassociator for anomaly detection and are in the process of testing the system at three industrial and commercial sites.

## 1  Introduction

An unexpected breakdown of an electric induction motor can cause financial loss significantly in excess of the cost of the motor. For example, the breakdown of a motor in a production line during a production run can cause the loss of work in progress as well as loss of production time.

When a motor does fail, it is not uncommon to replace it with an oversized motor based on the assumption that if a motor is not running at its design limit then it will survive longer. While this is frequently effective, this leads to significantly lower operating efficiencies and higher initial and operating costs.

The primary motivation behind this project is the observation that if a motor breakdown and be predicted before the actual breakdown occurs, then the motor can be replaced in a more orderly way, with minimal interruption of the process in which it is involved. The goal is to produce a system that is conceptually similar to a fuel gauge on an automobile. When the system detects conditions that indicate that the motor is approaching its end-of-life, the operators are notified that a replacement is necessary in the near future.

## 2    Background

At present, motors in critical operations that are subject to mechanical failures - for example, fire pump motors on US Navy vessels - are typically monitored by a human expert who periodically listens to the vibrations of the motor and, based on experience, determines whether the motor sounds healthy or sounds like a problem is developing. Since mechanical problems in motors typically lead to increased or changed vibrations, this technique can work well. Unfortunately, it depends on a competent and expensive expert.

In an attempt to automate motor monitoring, several vendors have "automated motor monitoring" equipment available. For mechanical failure monitoring, such systems typically rely on several accelerometers to measure the vibration of the motor at various points and along various axes. The systems then display information, primarily about the vibration spectrum, to an operator who determines whether the motor is functioning properly. These systems are expensive since they rely on several accelerometers, each of which is itself expensive, as well as data collection hardware and a computer. Further, the systems require an expert operator and frequently require that the motor be tested only when it is driving a known load.

Neither the human motor expert nor the existing motor monitoring systems provide an affordable solution for continuous on-line mechanical failure monitoring. However, the success of the human expert and existing vibration monitors does demonstrate that in fact, there is sufficient information in the vibration of an electric induction motor to detect imminent mechanical failures.

Siemens Energy and Automation has proposed a new product, the Siemens Advanced Motor Master System II (SAMMS II), that will continuously monitor and protect an electric induction motor while it is operating on-line. Like the presently available SAMMS, the SAMMS II is designed to provide protection against thermal and electrical overload an, in addition, it will provide detection of insulation deterioration and mechanical fault monitoring.

In contrast to existing systems and techniques, the SAMMS II is designed to (1) require no human expert to determine if a motor is developing problems; (2) be inexpensive; and (3) provide continuous, on-line monitoring of the motor in normal operation.

The requirements for the SAMMS II, in particular the cost constraint, require that several issues be resolved. First, in order to produce a low cost system, it is necessary to eliminate the need for expensive accelerometers. Second, wiring should be limited to the motor control center, i.e., it should not be necessary to run new signal wires from the motor control center to the motor. Third, the SAMMS II is to provide continuous on-line monitoring, so the system must adapt to or factor out the effect of changing loads on the motor. Finally since the SAMMS II would not necessarily be bundled with a motor and so might be used to control and monitor an arbitrary motor from an arbitrary manufacturer, the design can not assume that a full description of the motor construction is available.

## 3    Approach

The first task was to determine how to eliminate the accelerometers. Based on work done elsewhere (Schoen, Habetler & Bartheld, 1994), SE&A determined that it might be possible to use measurements of the current on a single phase of the power supply to estimate the vibration of the motor. This depends on the assumption that any vibration of the motor will cause the rotor to move radially relative to the stator which will cause changes in the airgap which, in turn, will induce changes in the current.

Experiments were done at the Georgia Institute of Technology to determine the feasibility of this idea using the same sort of data collection system described later. Early experiments indicated that, for a single motor driving a variety of loads, it is possible to distinguish

Table 1: Loads for motors #1 and #2.

| Load type | Load Magnitude |
|---|---|
| constant | half and full rated |
| sinusoidal oscillation at rotating frequency | half and full rated |
| sinusoidal oscillation at twice the rotating frequency | full rated |
| switching load (50% duty cycle) at rotating frequency | full rated |
| sinusoidal oscillation 28 Hz | half and full rated |
| sinusoidal oscillation at 30 Hz | full rated |
| switching load (50% duty cycle) at 30 Hz | full rated |

Table 2: Neural network classifier experiment.

| Features ($N$) | 48 | 63 | 64 | 110 | 320 |
|---|---|---|---|---|---|
| Performance on motor #1 | 100% | 100% | 92% | 100% | 100% |
| Performance on motor #2 | — | 30% | 25% | 55% | 37% |

between a current spectrum obtained from the motor while it is healthy and another obtained when the motor contains a fault. Moreover, it is also possible to automatically generate a classifiers that correctly determine the presence or absence of a fault in the motor.

The first, obvious approach to this monitoring task would seem to be to build a classifier that would be used to distinguish between a healthy motor and one that has developed a fault that is likely to lead to a breakdown. Unfortunately, this approach does not work.

As described above, we have successfully built classifiers of various sorts using manual and automatic techniques to distinguish between current spectra obtained from a motor when it is healthy and those obtained when it contains a fault.

However, since the SAMMS II will be connected to a motor before it fails and will be asked to identify a failure without ever seeing a labeled example of a failure from that motor, a classifier can only be used if it can be trained on data collected from one or more motors and then used to monitor the motor of interest. Unfortunately, experiments indicate that this will not work.

One of these experiments is illustrated in table 2. Several feedforward neural network classifiers were trained using examples from a single motor under four conditions: (1) healthy, (2) unbalanced, (3) containing a broken rotor bar and (4) containing a hole in the outer bearing race. The ten different loads listed in table 1 were applied to the motor for each of these conditions.

The networks contained $N$ inputs (where $N$ is given in table 2); 9 hidden units and 4 outputs. There were 40 training examples where each example is the average of 50 distinct magnitude scaled FFTs obtained from motor #1 from a single load/fault combination. The test data for which the results are reported in the table consisted of 40 averaged FFTs from motor #1 and 20 averaged FFTs (balanced and unbalanced only) from motor #2. The test set for motor #1 is completely distinct from the training set.

In the case where $n = 110$, the FFT components were selected to include the frequencies identified by the theory of motor physics as interesting for the three fault conditions and exclude all other components. This led to an improvement over the other cases where a single contiguous set of components was chosen, but the performance still degrades to about random chance instead of 100%.

This experiment clearly illustrates that is is possible to distinguish between healthy and faulty spectra obtained from the same motor. However, it also clearly illustrates that a

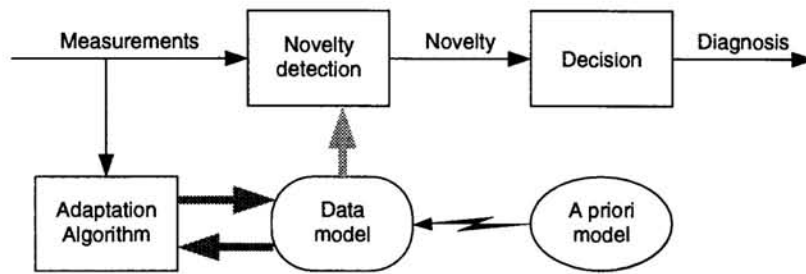

Figure 1: The basic form of an anomaly detection system.

classifier trained on one motor does not perform well on another motor since the error rates increase immensely. Based on results such as these, we have concluded that it is not feasible to build a single classifier that would be trained once and then placed in the field to monitor a motor. Instead we are pursuing an alternative based on anomaly detection which adapts a monitor to the particular motor for which it is responsible.

## 4  Anomaly detection

The basic notion of anomaly detection for monitoring is illustrated in figure 1. Statistical anomaly detection centers around a model of the data that was seen while the motor was operating normally. This model is produced by collecting spectra from the motor while it is operating normally. Once trained, the system compares each new spectrum to the model to determine how similar to or different from the training set it is. This similarity is described by an "anomaly metric" which, in the simplest case, can be thresholded to determine whether the motor is still normal or has developed a fault. Once the "anomaly metric" has been generated, various statistical techniques can be used to determine if there has been a change in the distribution of values.

## 5  A Neural Network-based Anomaly Detector

The core of the most successful monitoring system we have built to date is a neural network designed to function as an autoassociator (Rumelhart, Hinton & Williams, 1986, called it an "encoder"). We use a simple three layer feedforward network with $N$ inputs, $N$ outputs and $K < N$ hidden units. The input layer is fully connected to the hidden layer which is fully connected to the output layer. Each unit in the hidden and output layers computes $x_i = \sigma \left( \sum_{j=0}^{M_i} w_{i,j} x_j \right)$, where $x_i$ is the output of neuron $i$ which receives inputs from $M_i$ other neurons and $w_{i,j}$ is the weight on the connection from neuron $j$ to neuron $i$. The network is trained using the backpropagation algorithm to reconstruct the input vector on the output units. Specifically, if $x_i$ is one of $n$ input vectors and $\hat{x}_i$ is the corresponding output vector, the network is trained to minimize the sum of squared errors $E = \sum_{i=1}^{n} \|x_i - \hat{x}_i\|^2$. Once training is complete, the anomaly metric is $m_i = \|x_i - \hat{x}_i\|^2$.

## 6  Anomaly Detection Test

We have tested the effectiveness of the neural network autoassociator as an anomaly detector on several motors. For all these tests, the autoasociator had 20 hidden units. The hidden layer size was chosen after some experimentation and data analysis on motor #1, but no attempt was made to tune the hidden layer size for motor #2 or motor #3.

Motor #1 was tested using the ten different loads listed in table 1 and four different

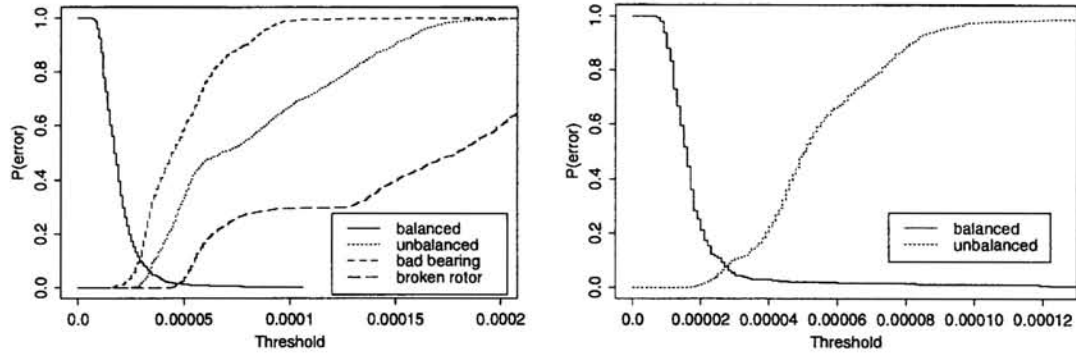

Figure 2: Probability of error as a function of threshold using individual FFTs on (a) motor #1 with 319 inputs and (b) motor #2 with 320 inputs.

health/fault conditions: healthy (balanced); unbalanced; broken rotor bar; and a hole in the outer bearing race. Motor #2 was tested while driving the same ten loads, but for one healthy and one faulty condition: healthy (balanced) and unbalanced.

For both motors #1 and #2, recordings of a single current phase were made as follows. For each fault condition, a load was selected and applied and the motor was run and the current signal recorded for five minutes. Then a new load was introduced and the motor was run again. The load was constant during any five minute recording session.

Motor #3 was tested using thirteen different loads, but only two fault conditions: healthy (balanced) and unbalanced. In this case, however, load changes occurred at random times. We preprocessed this data to to identify where the load changes occurred to generate the training set and the healthy motor test sets.

## 6.1  Preprocessing

Recordings were made on a digital audio tape (DAT). The current on a single phase was measured with a current transformer, amplified, notch filtered to reduce the magnitude of the 60Hz component, amplified again and then applied as input to the DAT. The notch filter was a switched capacitor filter which reduced the magnitude at 60Hz by about 30dB.

The time series obtained from the DAT was processed to reduce the sampling rate and then dividing the data into non-overlapping blocks and computing the FFT of each block. A subset of the FFT magnitude coefficients was selected and for each FFT, independent of any other FFT, the components were linearly scaled and translated to the interval $[\varepsilon, 1 - \varepsilon]$ (typically $\varepsilon = 0.02$). That is, for each FFT consisting of coefficients $f_0, \ldots, f_{n-1}$, we selected a subset, $\mathcal{F}$, (the same for all FFTs) of the components and computed $a = (1 - 2\varepsilon)(\max_{i \in \mathcal{F}} f_i - \min_{i \in \mathcal{F}} f_i)^{-1}$ and $b = \min_{i \in \mathcal{F}} f_i$. Then the input vector, x, to the network is $x_j = a(f_{i_j} - b) + \varepsilon$ where, for all $j < k$: $i_j, i_k \in \mathcal{F}$ and $i_j < i_k$.

## 6.2  Experimental Results

In figure 2a, we illustrate the results of a typical anomaly detection experiment on motor #1 using an autoassociator with 319 inputs and 20 hidden units. This graph illustrates the performance (false alarm and miss rates) of a very simple anomaly detection system which thresholds the anomaly metric to determine if the motor is good or bad. The decreasing curve that starts at threshold = 0, P(error) = 1 is the false alarm rate as a function of the threshold. Each increasing curve is the miss rate for a particular fault type.

In figure 2b we illustrate the performance of an autoassociator on motor #2 using an

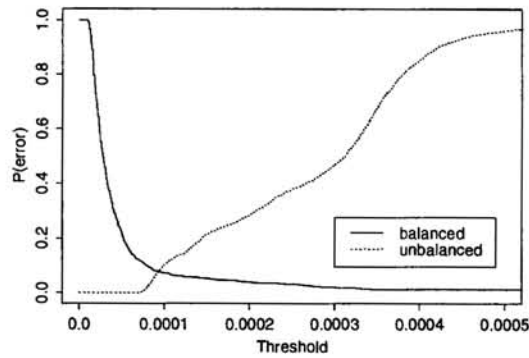

Figure 3: Probability of error for motor #3 using individual FFTs and 319 inputs.

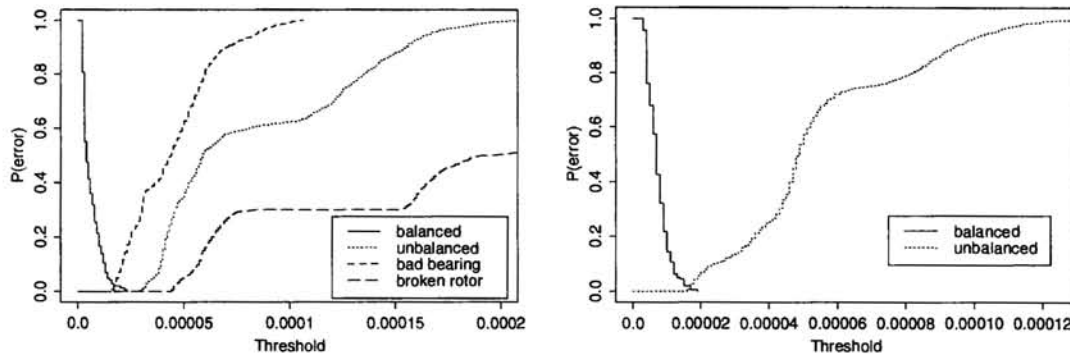

Figure 4: Probability of error using averaged FFTs for (a) motor #1 and 319 inputs (b) motor #2 and 320 inputs.

autoassociator with 320 inputs and 20 hidden units. Figure 3 shows our results on motor #3 using an autoassociator with 319 inputs.

We have found significant performance improvements by averaging several consecutive FFTs. In figure 4 we show the results for motors #1 and #2 when we averaged 11 FFTs to produce the input features. Compare these curves to those in figure 2. In particular, notice that the probability of error is much lower for the averaged FFTs when the good motor curve crosses any one of the faulty motor curves.

## 7   Candor System Design

Based on our experiments with autoassociators, we designed a prototype mechanical motor condition monitoring system. The functional system architecture is shown in figure 5. In order to control costs, the system is implemented on a PC. The system is designed so that each PC can monitor up to 128 motors using one 16-bit analog to digital converter. The signals are collected, filtered and multiplexed on custom external signal processing cards. Each card supports up to eight motors (with up to 16 cards per PC).

The system records current measurements from one motor at a time. For each motor, measurements are collected, four FFTs are computed on non-overlapping time series, and the four FFTs are averaged to produce a vector that is input to the neural network. The system reports that a motor is bad only if more than five of the last ten averaged FFTs produced an anomaly metric more than five standard deviations greater than the mean metric computed on the training set. Otherwise the motor is reported to be normal. In addition to monitoring the motors, the prototype systems are designed to record all measurements on tape to support

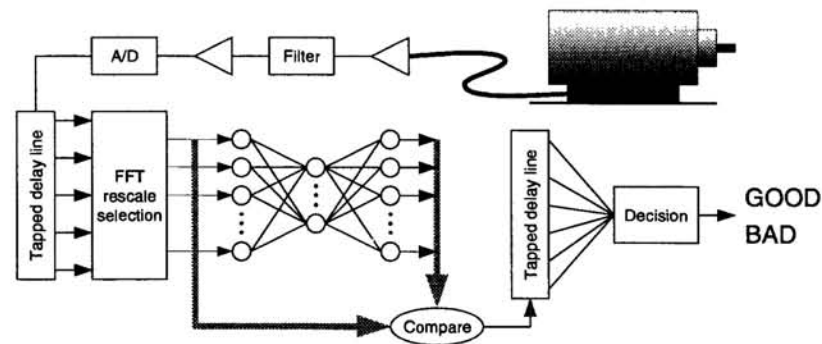

Figure 5: Functional architecture of Candor.

future experiments with alternative algorithms and tuning to improve performance.

To date, three monitoring systems have been installed: in an oil refinery, in a testing laboratory and on an office building ventilation system. The system has correctly detected the only failure it has seen so far: when a filter on the inlet to a water circulation pump became clogged the spectrum changed so much that the average daily novelty metric jumped from less than one standard deviation above the training set average to more than twenty standard deviations. We hope to have further test results in a year or so.

## 8   Related work

Gluck and Myers (1993) proposed a model of learning in the hippocampus based in part on an autoassociator which is used to detect novel stimuli and to compress the representation of the stimuli. This model has accurately predicted many of the classical conditioning behaviors that have been observed in normal and hippocampal-damaged animals. Based on this work, Japkowicz, Myers and Gluck (1995) independently derived an autoassociator-based novelty detector for machine learning tasks similar to that used in our system.

Together with Gluck, we have tested an autoassociator based anomaly detector on helicopter gearbox failures for the US Navy. In this case, the autoassociator is given 512 inputs consisting of 64 vibration based features from each of 8 accelerometers mounted at different locations on the gearbox. In a blind test, the autoassociator was able to correctly distinguish between feature vectors taken from a damaged gearbox and other feature vectors taken from normal gearboxes, all recorded in flight. Our anomaly detector will be included in test flights of a gearbox monitoring system later this year.

## References

Gluck, M. A. & Myers, C. E. (1993). Hippocampal mediation of stimulus representation: A compuational theory. *Hippocampus*, *3*(4), 491–561.

Japkowicz, N., Myers, C., & Gluck, M. A. (1995). A novelty detection approach to classification. In *Proceedings of the Fourteenth International Joint Conference on Artificial Intelligence*.

Rumelhart, D., Hinton, G., & Williams, R. (1986). Learning internal representations by error propagation. In D. Rumelhart & J. McClelland (Eds.), *Parallel Distributed Processing* (pp. 318–362). MIT Press.

Schoen, R., Habetler, T., & Bartheld, R. (1994). Motor bearing damage detection using stator current monitoring. In *Proceedings of the IEEE IAS Annual Meeting*.
